# Unsupervised Detection of Regions of Interest Using Iterative Link Analysis

**Gunhee Kim**
School of Computer Science
Carnegie Mellon University
gunhee@cs.cmu.edu

**Antonio Torralba**
Computer Science and Artificial Intelligence Laboratory
Massachusetts Institute of Technology
torralba@csail.mit.edu

## Abstract

This paper proposes a fast and scalable alternating optimization technique to detect regions of interest (ROIs) in cluttered Web images without labels. The proposed approach discovers highly probable regions of object instances by iteratively repeating the following two functions: (1) choose the exemplar set (*i.e.* a small number of highly ranked reference ROIs) across the dataset and (2) refine the ROIs of each image with respect to the exemplar set. These two subproblems are formulated as ranking in two different similarity networks of ROI hypotheses by link analysis. The experiments with the PASCAL 06 dataset show that our unsupervised localization performance is better than one of state-of-the-art techniques and comparable to supervised methods. Also, we test the scalability of our approach with five objects in Flickr dataset consisting of more than 200K images.

## 1   Introduction

This paper proposes an unsupervised approach to the detection of regions of interest (ROIs) from a Web-sized dataset (Fig.1). We define the regions of interest as highly probable rectangular regions of object instances in the images. The extraction of ROIs is extremely helpful for recognition and Web user interfaces. For example, [3, 5] showed comparative studies in which ROI detection is useful to learn more accurate models, which leads to nontrivial improvement of classification and localization performance. In the recognition of indoor scenes [17], the local regions that contain objects may have special meaning to characterize the scene description. Also, many Web applications allow a user to attach notes on user-specified regions in a cluttered image (*e.g.* Flickr Notes). Our algorithm can make this cumbersome annotation easier by suggesting the regions a user may be interested in.

Our solution to the problem of unsupervised ROI detection is inspired by an alternating optimization. Alternating optimization is one of widely used heuristics where optimization over two sets of variables is not straightforward, but optimization with respect to one while keeping the other fixed is much easier and solvable. This approach has been successful in a wide range of areas such as K-means, Expectation-Maximization, and Iterative Closest Point algorithms [2].

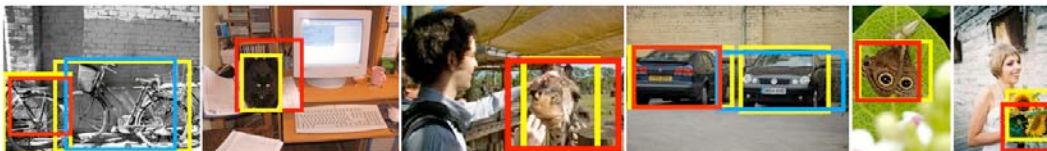

Figure 1: Detection of regions of interest (ROIs). Given a Web-sized dataset, our algorithm detects bounding box-shaped ROIs that are statistically significant across the dataset in an unsupervised manner. The yellow boxes are groundtruth labels, and the red and blue ones are ROIs detected by the proposed method.

The unsupervised ROI detection can be though of as a chicken-and-egg problem between (1) finding exemplars of objects in the dataset and (2) localizing object instances in each image. If class-representative exemplars are given, the detection of objects in images is solvable (*i.e.* a conventional *detection* or *localization* problem). Conversely, if object instances are clearly annotated beforehand, the exemplars can be easily obtained (*i.e.* a conventional *modeling* or *ranking* problem).

Given an image set, first we assume that each image itself is the best ROI (*i.e.* the most confident object region). Then a small number of highly ranked ones among the selected ROIs are chosen as exemplars (called *hub seeking*), which serve as references to refine the ROIs of each image (called *ROI refinement*). We repeat these two updates until convergence. The two steps are formulated as ranking in two different similarity networks of ROI hypotheses by link analysis. The *hub seeking* corresponds to finding a central and diverse hub set in a network of the selected ROIs (*i.e.* inter-image level). The *ROI refinement* is the ranking in a bipartite graph between the hub sets and all possible ROI hypotheses of each image (*i.e.* intra-image level).

Our work is closely related to topics on *ROI detection* [3, 5, 17, 14], *unsupervised localization* [9, 24, 21, 18, 1, 12], and *online image collection* [13, 19, 6]. The *ROI detection* and *unsupervised localization* share a similar goal of detecting the regions of objects in cluttered images. However, most previous work has been successful for standard datasets with thousands of images. On the other hand, our goal is to propose a simple and fast method that can take advantage of enormous amounts of Web data. The main objective of *online image collection* is to collect relevant images from highly noisy data queried by keywords from the Web. Its main limitation is that much of the previous work requires additional assumptions such as a small number of seed images in the beginning [13], texts and HTML tags associated with images [19], and user-labeled images [6]. On the other hand, no additional meta-data are required in our approach.

Recently, link analysis techniques on visual similarity networks were successfully exploited in computer vision problems [12, 15, 11, 16]. [15] applied the *random walk with restart* technique to the auto-captioning task. However, their work is a supervised method requiring annotated caption words for the segmented regions in training images. [12] is similar to ours in that the unsupervised classification and localization are the main objectives. However, their method suffers from a scalability issue, and thus their experiments were performed using only 600 images. [11] successfully applied the *PageRank* technique to a large-scale image search, but unlike ours their approach is evaluated with quite clean images and sub-image level localization is not dealt with. Likewise, [16] also exploited the matching graph of a large-scale image set, but the localization was not discussed.

The main advantages of our approach are summarized as follows. First, the proposed method is extremely simple and fast, with compelling performance. Our approach shows superior results over a state-of-the-art unsupervised localization method [18] for the PASCAL 06 dataset. We proposed a simple heuristic for scalability to make the computation time linear with the data size without severe performance drop. For example, the localization of 200K images took only 4.5 hours with naive matlab implementation on a single PC equipped with Intel Xeon 2.83 GHz CPU (once image segmentation and feature extraction were done). Second, our approach is *dynamic* thanks to the *evolving network* representation. At every iteration, new ROI hypothesis are added and trivial ones are removed from the network while reusing a large portion of previously computed information. Third, unlike most previous work, our approach requires neither human annotation, meta-data, nor initial seed images. Finally, we evaluate our approach with a challenging Flickr dataset of up to 200K images. Although some work [22] in image retrieval uses millions of images, this work has a different goal from ours. The objective of image retrieval is to quickly index and search the nearest images to a given query. On the other hand, our goal is to localize objects in every single image of a dataset without supervision.

## 2 ROI Candidates and Description

The input to our algorithm is a set of images $\mathcal{I} = \{I_1, I_2, ..., I_{|\mathcal{I}|}\}$. The first task is to define a set of ROI hypotheses from the image set $\mathcal{R} = \{R_1, R_2, ..., R_{|\mathcal{I}|}\}$. Ideally, the set of ROI hypotheses $R_a = \{r_{a1}, ..., r_{am}\}$ of an image $I_a$ enumerates all plausible bounding boxes, and at least one of them is supposed to be a good object annotation. Fig.2 shows the procedure of ROI hypothesis generation. Given an image, 15 segments are extracted by Normalized cuts [20]. The minimum rectangle to enclose each segment is defined as initial ROI hypotheses. Since the over-segmentation

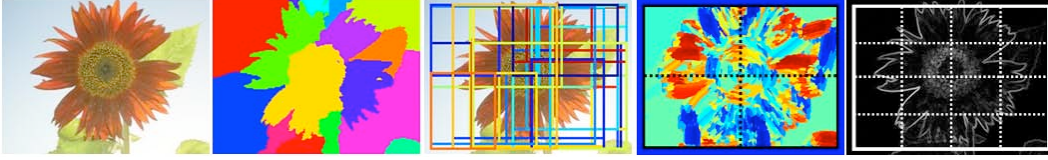

Figure 2: An example of ROI extraction and description. From left to right: (a) An input image. (b) 15 segments. (c) 43 ROI hypotheses. (d) Distribution of visual words. (e) Edge gradients.

is unavoidable in most cases, the combinations of the initial hypotheses are also considered. We first compute pairwise minimum paths between the initial hypotheses using the Dijkstra algorithm. Then the bounding boxes to enclose those minimum paths are added to the ROI hypothesis set. Finally, a largely overlapped pair of ROIs is merged if $\frac{r_{ai} \cap r_{aj}}{r_{ai} \cup r_{aj}} > 0.8$. Note that the hypothesis set always includes the image itself as the largest candidate, and the average set size is about 50.

Each ROI hypothesis is represented by two types of descriptors, which are spatial pyramids of visual words [17] and HOG [3]. As usual, the visual words are generated by vector quantization to randomly selected SIFT descriptors. K-means is applied to form a dictionary of 200 visual words. A visual word is assigned to each pixel of an image by finding nearest cluster center in the dictionary, and then binned using a two-level spatial pyramid. The oriented gradients are computed by Canny edge detection and Sobel mask. Then the HOG descriptor is discretized into 20 orientation bins in the range of $[0°,180°]$ by following [3]. The pyramid level is up to three. The similarity measure between a pair of ROIs is cosine similarity, which is simply calculated by dot product of two $L_2$ normalized histograms. Here both descriptors are equally weighted.

## 3 The Algorithm

### 3.1 Similarity Networks and Link Analysis Techniques

All inferences in our approach are based on the link analysis of $k$-nearest neighbor similarity network between ROI hypotheses. The similarity network is a weighted graph $\mathcal{G} = (\mathcal{V}, \mathcal{E}, \mathcal{W})$, where $\mathcal{V}$ is the set of vertices that are ROI hypotheses. $\mathcal{E}$ and $\mathcal{W}$ are edge and weight sets discovered by the similarity measure in the previous section. Each vertex is only connected to its $k$-nearest neighbors with $k = a \cdot \log |\mathcal{V}|$ [23], where $a$ is a constant set to 10. It results in a sparse network, which is more advantageous in terms of computational speed and accuracy. It guarantees that the complexity of network analysis is $\mathcal{O}(|\mathcal{V}| \log |\mathcal{V}|)$ at worst. Finally, the network is row normalized so that the edge weight from note $i$ and $j$ indicates the probability of a random surfer jumping from $i$ to $j$. The link analysis technique we use is *PageRank* [4, 10]. Given a similarity matrix $\boldsymbol{G}$, it computes the same length of *PageRank* vector $\boldsymbol{p}$, which assigns a ranked score to each vertex of the network. Intuitively, the *PageRank* scores of the network of ROI hypotheses are indices of the goodness of hypotheses.

### 3.2 Overview of the Algorithm

Algorithm 1 summarizes the proposed algorithm. The main input is the set of ROI hypotheses $\mathcal{R}$ generated by the method of section 2. The output is the set of selected ROIs $\mathcal{S}^*(\subset \mathcal{R})$. In each image, usually one or two, and rarely more than three, of the most promising ROIs are chosen.

The basic idea of our approach is to jointly optimize the ROI selection of each image and the examplar detection among the selected ROIs. Examplars correspond to hubs in our network representation. We begin with images themselves as an initial set of ROI selection $\mathcal{S}^{(0)}$ (**Step 1**). Even though this initialization is quite poor, highly ranked hubs among the ROIs are likely to be much more reliable. They are detected by the function Hub seeking (**Step 3**). Then, the hub sets are exploited to refine the ROIs of each images by the function Hub seeking (**Step 5**). In turn, those refined ROIs are likely to lead to a better hub set at the next iteration. The alternating iterations of those two functions are expected to lead convergence for not only the best ROI selection of each image but also the most representative ROIs of the data set. An example of evolution of ROI selection is shown in Fig.4.(c). Although our algorithm forces to select at least one ROI for each image, the *PageRank* vector by Hub seeking can indicate the confidence of each ROI, which can be used to filter out wrongly selected ROIs. Conceptually, both functions share a similar ranking problem to

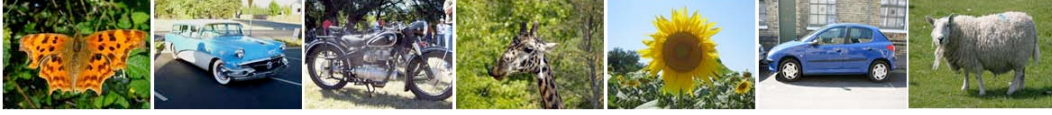

Figure 3: Examples of hub images. The pictures illustrate highest-ranked images in 10,000 randomly selected images from five objects of our Flickr dataset and all {train+val}images from two objects of the PASCAL06.

select a small subset of highly ranked nodes from the input networks of ROI hypotheses. They will be discussed in the following subsections in detail.

Inherently, a good initialization is essential for alternating optimization. Our key assumption is as follows: *Provided that the similarity network includes a sufficiently large number of images, the hub images are likely to be good references*. This is based on the finding of our previous work [12]: If each visual entity votes for others that are similar to itself, this democratic voting can reveal the dominant statistics of the image set. Although the images in a dataset are highly variable, the more repetitive visual information may get more similarity votes, which can be easily and quickly discovered as hubs by link analysis. Fig.3 supports this argument in our dataset. It illustrates top-ranked images of our dataset in which the objects are clearly shown in the center with significant size. Obviously, they are excellent initialization candidates.

Since we deal with discrete patches from unordered natural images on the Web, it is extremely difficult to analytically understand several important behaviors of our algorithm such as convexity, convergence, sensitivity to initial guess, and quality of our solution. One widely used assumption in the optimization with image patches is linearity with small incremental displacement (*e.g.* AAM [8]). However, it is not the case in our problem and causes severe computation increase. These issues may be open challenges for the optimization of large-scale image analysis.

---

**Algorithm 1** The Algorithm

---

**Input:** ROI hypothesis $\mathcal{R}$ associated with image set $\mathcal{I}$.
**Output:** The set of selected ROIs $\mathcal{S}^*(\subset \mathcal{R})$, where $\mathcal{S}^* = \mathcal{S}^{(T)}$ when converged at $T$.
  1: $\mathcal{S}^{(0)} \leftarrow$ largest ROI hypothesis in each image.
  **while** $\mathcal{S}^{(t-1)} \neq \mathcal{S}^{(t)}$ **do**
    2: Generate $k$-NN similarity network $\boldsymbol{G}^{(t)}$ of $\mathcal{S}^{(t)}$.
    3: $\mathcal{H}^{(t)} \leftarrow$ Hub seeking($\boldsymbol{G}^{(t)}$), where the hub set $\mathcal{H}^{(t)} \subset \mathcal{S}^{(t)}$
    **for all** $I_a \in \mathcal{I}$ unless ROI selection of $I_a$ is not changed for several consecutive times **do**
      4: $s_a^{(t)} \leftarrow$ ROI refinement($\mathcal{H}^{(t)}, R_a$), where $s_a^{(t)}$: ROI selection of $I_a$, $R_a$: ROI hypotheses of $I_a$.
      5: $\mathcal{S}^{(t)} \leftarrow \mathcal{S}^{(t)} \cup s_a^{(t)} \backslash s_a^{(t-1)}$ .
    **end for**
  **end while**

---

**Algorithm 2** Hub seeking function

---

**Input:** (1) Network $\boldsymbol{G}^{(t)}$. (2) Window size: $d$.
**Output:** (1) Hub set $\mathcal{H}^{(t)}$.
  1: Compute *PageRank* vector $\boldsymbol{p}$ of $\boldsymbol{G}^{(t)}$.
  **for all** vertex $v \in \boldsymbol{G}^{(t)}$ **do**
    2: Find the neighbor set of $v$ $\mathcal{N}_v = \{u |$ max reachable probability from $v$ to $u > d\}$.
    3: Find local maxima node of $v$ $\boldsymbol{m}(v) = \arg\max_u \boldsymbol{p}(\mathcal{N}_v)$ where $u \in \mathcal{N}_v$.
    4: $\mathcal{H}^{(t)} \leftarrow v$ if $v = \boldsymbol{m}(v)$.
  **end for**

---

**Algorithm 3** ROI refinement function

---

**Input:** (1) Hub set $\mathcal{H}^{(t)}$. (2) $R_a$, ROI hypotheses of $I_a$
**Output:** (1) The selected ROIs $s_a^{(t)}(\subset R_a)$.
  1: Generate $k$-NN self-similarity matrix $\boldsymbol{W}_i$ of $R_a$ and $k$-NN similarity matrix $\boldsymbol{W}_o$ between $R_a$ and $\mathcal{H}^{(t)}$. Both of them are row-normalized.
  2: Generate augmented bipartite graph $\boldsymbol{W} = \begin{pmatrix} \alpha \boldsymbol{W}_i & (1-\alpha)\boldsymbol{W}_o \\ \boldsymbol{W}_o^{\mathrm{T}} & \boldsymbol{0} \end{pmatrix}$
  3: Compute *PageRank* vector $\boldsymbol{p}$ of $\boldsymbol{W}$.
  4: $s_a^* = \arg\max_{r_{aj}} \boldsymbol{p}(r_{aj})$ where $r_{aj} \in \mathcal{R}_a$.

---

### 3.3 Hub Seeking with Centrality and Diversity

The goal of this step is to detect a hub set $\mathcal{H}^{(t)}$ from $\mathcal{S}^{(t)}$ by analyzing the network $\boldsymbol{G}^{(t)}$. The main criteria are *centrality* and *diversity*. In other words, the selected hub set should be not only highly ranked but also diverse enough not to lose various aspects of an object. To meet this requirement, we design the hub seeking inspired by Mean Shift [7]. Given feature points, the algorithm creates a fixed-radius window at each point. Then each window iteratively moves into the direction of

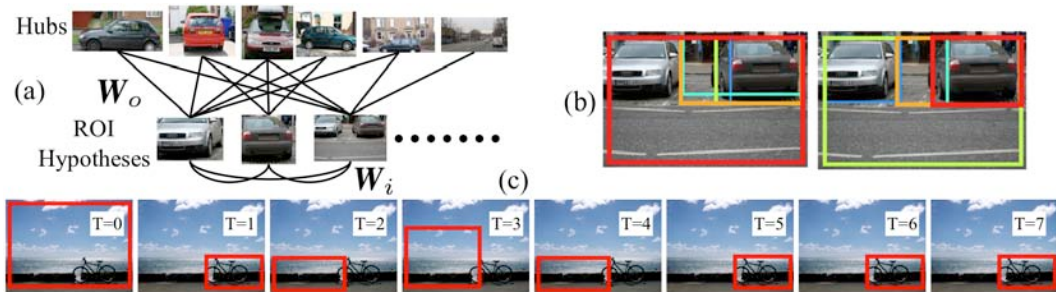

Figure 4: (a) An example of a bipartite graph between the hub set and ROI hypotheses of an image. The similarity between hubs and hypotheses is captured by $W_o$ and the affinity between the hypotheses by $W_i$. The hub set is sorted by *PageRank* values from left and right. The values of leftmost and rightmost are 0.0081 and 0.0024, respectively. They successfully capture various aspects related to the *car* object. (b) The effect of the augmented bipartite graph. The left image is with $\alpha = 0$ and the right with $\alpha = 0.1$. The ranking of hypotheses is represented by jet colormap from red (high) to blue (low). In the left, the weights from the red box to the blue one are (0.052, 0.050, 0.049, 0.049, 0.049); in the right, (0.060, 0.060, 0.059, 0.059, 0.057). (c) An example of ROI evolution. At $T = 0$, the selected ROI is an image itself but is converged to the real object after $T = 5$.

the maximum increase in the local density function until it reaches a local maximum. Those local maxima become the modes, and the data points that converge to the same maxima are clustered.

The proposed algorithm 2 works in the same manner. For each vertex, we define the search window in the form of maximum reachable probability $d$ (**Step 2**). The window covers the vertices whose maximum reachable probability is larger than $d$. For example, given $d = 0.1$, $w_{ij} = 0.6$, $w_{jk} = 0.2$, the probability of vertices $i$ to $k$ is $0.6 \times 0.2 = 0.12 > d$. Then $k$ is considered inside the search window of $i$. For the density function, we use the *PageRank* vector because it is proportional to the vertex degree if the graph is symmetric and connected [25]. In **Step 3**, we compute the vector $\boldsymbol{m}$ that assigns the local maximum vertex within the window of each vertex in $\boldsymbol{G}^{(t)}$. If $v = \boldsymbol{m}(v)$, the $v$ is a local maximum, and it is added to $\mathcal{H}^{(t)}$. Additionally, we can easily perform the clustering from $\boldsymbol{m}$. For each node, the search window keeps moving the maximum direction indicated by $\boldsymbol{m}$ until it reaches the local maximum. Then the nodes that converge to the same maxima can be clustered.

### 3.4 ROI Refinement

Formally, this step is to define a nonparametric function for each image $f_a : R_a \rightarrow R^+$ (positive real number) with respect to the hub set $\mathcal{H}^{(t)}$. Then the hypothesis with maximum ranked value is chosen as the best ROI. In order to solve this problem, we first construct an augmented bipartite graph $\boldsymbol{W}$ between the hub set $\mathcal{H}^{(t)}$ and all possible ROIs $R_a$ as shown in **Step 2** of Algorithm 3 (see Fig.4(a)). For better understanding, let us first consider a pure bipartite graph with $\alpha = 0$. Then the matrix $\boldsymbol{W}$ represents the similarity voting between the ROI candidates and the hub set. If the *PageRank* vector $\boldsymbol{p}$ of $\boldsymbol{W}$ is computed, then $\boldsymbol{p}(R_a)$ summarizes the relative importance of each ROI hypothesis with respect to the $\mathcal{H}^{(t)}$, which is exactly what we require. Rather than a pure bipartite graph ($\alpha = 0$), we augment it by nonzero $\alpha$. Fig.4.(b) explains the effects of $\alpha$. The left image shows the result of $\alpha = 0$. Even though the red hypothesis is the maximum, several hypotheses near the dark gray car have significant values. With nonzero $\alpha = 0.1$, those hypotheses are allowed to augment each other, so the maximum ROI is changed to a hypothesis on the car. In terms of link analysis, if a random surfer visits nodes of ROI hypotheses ($R_a$), it jumps to other hypotheses with probability $\alpha$ or other hubs with $1 - \alpha$. Since the nearby hypotheses share large portions of rectangles, they have higher similarity, which results in more votes for nearby hypotheses.

### 3.5 Scalability Setting

The bottleneck of our approach is the **Step 3** of Algorithm 1. The network generation requires quadratic computation of cosine similarity of $\mathcal{S}^{(t)}$. In order to bound the computational complexity, we limit the maximum number of images to be considered each run of Algorithm 1 by constant number $N$. $N$ should be small enough not to suffer from computational burden. Simultaneously, it should be large enough to successfully detect the meaningful statistics from an extremely variable

dataset. (In experiments, $N$ is set to 10,000.) If the dataset size $|\mathcal{I}| > N$, we randomly sample $N$ images from $\mathcal{I}$ and construct initial consideration set $\mathcal{I}_c \subset \mathcal{I}$. Algorithm 1 is applied to the image set $\mathcal{I}_c$ to obtain $\mathcal{S}_c^*$. Then we generate new $\mathcal{I}_c$ by sampling unvisited images from $\mathcal{I}$. In order to reuse the result of $\mathcal{S}_c^*$ for the next iteration, we sample $x\%$ of $N$ from previous $\mathcal{S}_c^*$ based on the *PageRank* values of the network $\boldsymbol{G}^*$ of $\mathcal{S}_c^*$. In other words, the highly ranked (*i.e.* highly confident) ROIs in the previous step are reused to expedite the convergence of next iteration. We iterate the above strategy until all images are examined. This simple heuristic allows our technique to analyze an extremely large dataset in a linear time without significant performance drop.

## 4   Results

We evaluate our approach with two different experiments, (1) performance tests with PASCAL VOC 2006[1] and (2) scalability tests with Flickr images. The PASCAL dataset provides groundtruth labels, so our approach is quantitatively evaluated and compared with other approaches. Using Flickr dataset, we examine the scalability of our method in a real-world problem. The images are collected by a query that consists of one object word and one context word. We downloaded images of the objects {*butterfly+insect*(69,990), *classic+car*(265,731), *motorcycle+bike*(106,590), *sunflower*(165,235), *giraffe+zoo*(53,620)}. The numbers in parentheses are dataset sizes.

### 4.1   Performance Tests

The input of our algorithm consists of unlabeled images, which may include a single object (called as *weakly supervised*) or multiple objects (called *unsupervised*). For *unsupervised* cases, we perform not only localization but also classification according to object types. The PASCAL 06 dataset is so challenging to use that only very rare previous work has used it for unsupervised localization. For comparison, we ran publicly available code of one of the state-of-the-art techniques proposed by Russell et al[2] [18] in the identical setting.

The PASCAL dataset consists of {train+val+test}. However, our approach requires only images as an input, and thus all of the {train+val+test} images are used without discrimination between them. Note that our task is an image annotation not a learning problem that requires training and testing steps. The performance is measured by following the protocol of PASCAL evaluation: (1) The performance is evaluated from only the {test} set. In practice, there is very little performance difference between analysis of all {train+val+test} and {test} only. (2) The detection is considered correct if the overlap between the prediction and ground truth exceeds 50%.

**Weakly supervised localization.** Fig.5 shows the detection performance as Precision-Recall (PR) curves. For [18], we iterate experiments by changing the number of topics from two to six, and the best results are reported. For clear comparison between our results and [18], we select only the best bounding box in each image. We also present the best result of each object in VOC06 competition. Strictly speaking, it is not a valid comparison because the experimental setups of VOC06 competition and ours are totally different. However, we illustrate them as references to show how closely our approach can reach the best supervised methods in VOC 06 for the localization. Although the performance varies according to objects, our approach significantly outperformed [18] except in *cow*. Promisingly, the performances of our approach for *bicycle* and *motorbike* are comparable, and those for *bus*, *cat*, and *dog* objects are superior to the bests of the supervised methods in VOC06.

**Unsupervised classification and localization.** Here we evaluate how well our approach works for unsupervised classification and localization tasks (*i.e.* images of multiple objects without any annotation are given). Since both our method and [18] aim at sub-image level classification and detection, we first find out the most confident region of each image, and run the clustering by LDA in [18] and spectral clustering [20] in our method. The evaluation of classification follows the rule of VOC06 by the ROC curves as shown in Fig.6. We also show the best of the VOC06 submissions for supervised classification as a reference. As shown in Fig.6.(a)−(c), our method and [18] present similar ROC performance. In other words, both methods are quite good at ranking for classification. However, the classification rates of our method are better by about 10% for both 3-object and 4-object cases. (Ours: **69.08**%; [18]: 59.05% for {$bicycle, car, dog$}. Ours: **59.51**%; [18]: 50.99%

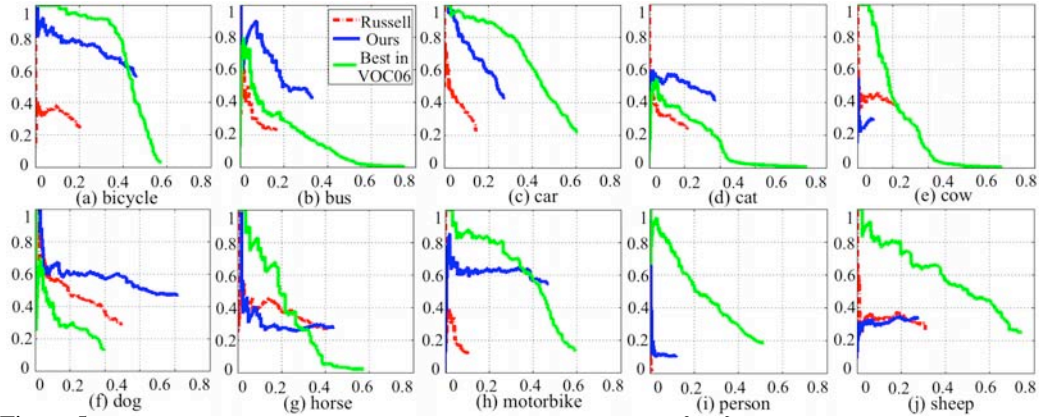

Figure 5: Results of weakly supervised localization. PR curves for the {test} sets of all objects in the PASCAL 06 dataset. (Ours: blue; [18]: red; the best of VOC06: green). Note that our localization and that of [18] are unsupervised, but the VOC06 localization is supervised. (X-axis: recall; Y-axis: precision).

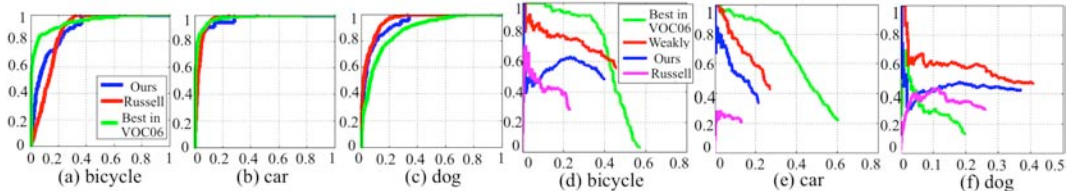

Figure 6: Results of unsupervised classification and localization. (a)−(c) ROC curves for {test} set of {bicycle, car, dog}. (Ours: blue; [18]: red; the best of VOC06: green). The AUCs of ours, [18], and the best of VOC06 are bicycle:(0.892, 0.869, 0.948), car:(0.968, 0.965, 0.977), and dog:(0.932, 0.954, 0.876), respectively. (X-axis: false positive rates, Y-axis: true positive rates). (d)−(f) PR curves for unsupervised localization of ours (blue) and [18] (magenta). For comparison, we also represent the results of our weakly supervised localization (red) and the best of VOC 06 (green). (X-axis: recall, Y-axis: precision).

for {bicycle, car, dog, sheep}.) We also show the unsupervised localization performance as PR-curves in Fig.6.(d)−(f). For comparison, we also represent the results of our weakly supervised experiments and the bests of VOC 06 for corresponding objects. The nontrivial performance drop is observed due to the classification errors and distraction by other objects in the dataset.

## 4.2 Scalability Tests

It is an open question how to evaluate the results of a large number of Web-downloaded images that have no ground-truth. For a quantitative evaluation, we manually annotated $0.5\%$ randomly selected images of datasets, and they are used as limited but approximate indices of performance measures. According to the data sizes used in experiments, we randomly pick $x\%$ from the annotated set and $(100 - x)\%$ from the non-annotated set. The $x$ is $\{20, 10, 5, 1, 0.5, 0.5\}$ for the dataset size of $\{500, 5K, 10K, 50K, 100K, 200K\}$.

**Weakly supervised localization.** One interesting question we address here is how performances and computation times vary as a function of data sizes. The experiments are repeated ten times for each dataset size, and the median (*i.e.* fifth-best) performance scores are reported. Similarly to previous tests, we select only the best ROI per image. As shown in Fig.7, the performances of 500 images fluctuate, but the results of the dataset size above 5K are stable. As the dataset size increases, a small performance improvement is observed. Since the maximum number of images at each running of the algorithm is bounded by $N (= 10, 000)$, the computation times are linear to the number of images, and the performances of the data size above $N$ are similar each other.

**Perturbation tests.** Here we test the goodness of selected ROIs from a different view: robustness of ROI detection against random network formation. For example, given an image $I_a$, we can generate 100 sets of 200 randomly selected images that include $I_a$. If the ROI selection for $I_a$ is repetitive across 100 different sets, we can say the ROI estimator for $I_a$ is confident. This procedure is similar to *bootstrapping* or *cross-validation*.

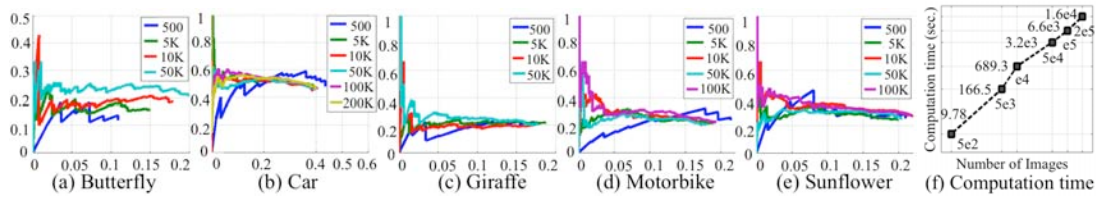

Figure 7: Weakly supervised localization. (a) PR curves for five objects of our Flickr dataset by varying dataset sizes from 500 to 200K. (b) The log-log plot between the number of images and computation times for the *car* object. The slope of each range is $\{1.23, 2.05, 0.95, 1.05, 1.28\}$ from left to right.

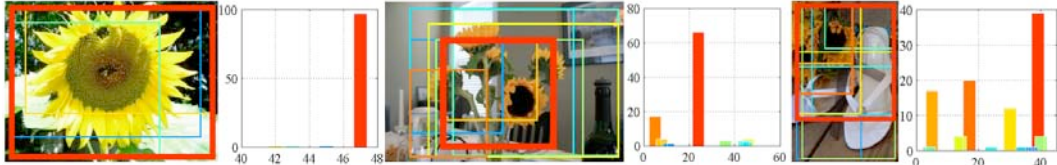

Figure 8: Examples of perturbation tests. The histograms summarize how many times each ROI is selected in 100 random sets. The frequencies of particular ROIs are represented by the thickness of bounding boxes and the jet colormap from red (high) to blue (low). From left to right, the entropies of the distributions are $\{0.2419, 1.6846, 2.4331\}$, respectively. (X-axis: ROI hypotheses; Y-axis: Frequency).

Fig.8 shows some examples of the perturbation tests. The histogram indicates how many times each ROI hypothesis is selected among 100 random sets. From left to right, one can see the increase of the difficulty of ROI detection. A peak is observed in an obvious image, but the distribution is wider in a challenging image. The entropy of the distribution can be an index of the measure of difficulty or the confidence of the estimator of the image.

**More localization examples.** Fig.9 shows more examples of localization in our approach. The third row illustrates some typical examples of failure. Frequently co-occurred objects can be detected instead such as flowers in *butterfly* images, insects on *sunflowers*, other animals in the zoo, and persons everywhere. Also, sometimes small multiple instances are detected by one ROI or a part of an object is discovered (*e.g.* a giraffe face rather than the whole body).

## 5   Discussion

We proposed an alternating optimization approach for scalable unsupervised ROI detection by analyzing the statistics of similarity links between ROI hypotheses. Both tests with PASCAL 06 and Flickr datasets showed that our approach is not only comparable to other unsupervised and supervised techniques but also applicable to real images on the Web.

**Acknowledgement.** Funding for this research was provided by NSF Career award (IIS 0747120).

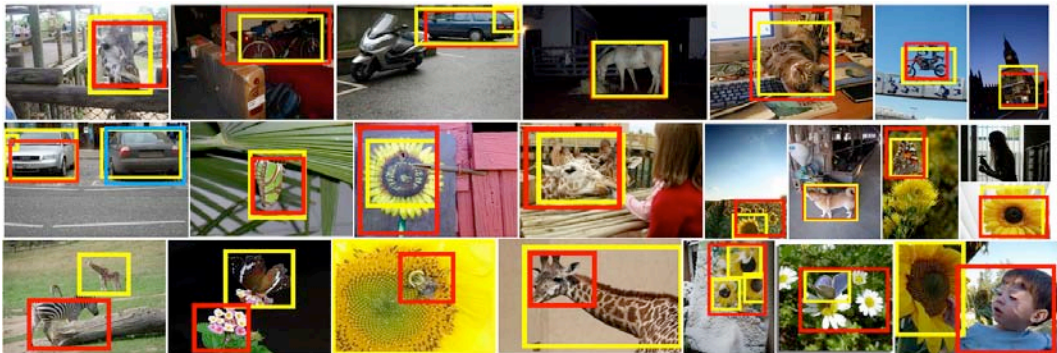

Figure 9: More examples of object localization. The first and second rows represent successful detection, and the third row illustrates some typical failures. The yellow boxes are groundtruth labels, and the red and blue ones are ROIs detected by the proposed method.

## Footnotes

[1]The dataset is available at http://www.pascal-network.org/challenges/VOC/

[2]The code is available at http://www.di.ens.fr/∼russell/projects/mult_seg_discovery/index.html

# References

[1] N. Ahuja and S. Todorovic. Learning the taxonomy and models of categories present in arbitrary images. In *ICCV*, 2007.

[2] P. J. Besl and N. D. McKay. A method for registration of 3-d shapes. *IEEE Trans. on Pattern Analysis and Machine Intelligence*, 14(2):239–256, 1992.

[3] A. Bosch, A. Zisserman, and X. Munoz. Image classification using random forests and ferns. In *ICCV*, 2007.

[4] S. Brin and L. Page. The anatomy of a large-scale hypertextual web search engine. In *WWW*, 1998.

[5] O. Chum and A. Zisserman. An exemplar model for learning object classes. In *CVPR*, 2007.

[6] B. Collins, J. Deng, K. Li, and L. Fei-Fei. Towards scalable dataset construction: An active learning approach. In *ECCV*, 2008.

[7] D. Comaniciu and P. Meer. Mean shift: A robutst approach toward feature space analysis. *IEEE Trans. on Pattern Analysis and Machine Intelligence*, 24(5):603–619, 2002.

[8] T. F. Cootes, G. J. Edwards, and C. J. Taylor. Active appearance models. *IEEE Trans. on Pattern Analysis and Machine Intelligence*, 23(6):681–685, 2001.

[9] R. Fergus, L. Fei-Fei, P. Perona, and A. Zisserman. Learning object categories from google's image search. In *ICCV*, pages 1816–1823, Oct. 2005.

[10] G. Jeh and J. Widom. Scaling personalized web search. In *WWW*, 2003.

[11] Y. Jing and S. Baluja. Visualrank, pagerank for google image search. *IEEE Trans. on Pattern Analysis and Machine Intelligence*, 30(11):1–31, 2008.

[12] G. Kim, C. Faloutsos, and M. Hebert. Unsupervised modeling of object categories using link analysis techniques. In *CVPR*, 2008.

[13] L.-J. Li, G. Wang, and L. Fei-Fei. Optimol: automatic object picture collection via incremental model learning. In *CVPR*, 2007.

[14] T. Liu, J. Sun, N.-N. Zheng, X. Tang, and H.-Y. Shum. Learning to detect a salient object. In *CVPR*, 2007.

[15] J.-Y. Pan, H.-J. Yang, C. Faloutsos, and P. Duygulu. Automatic multimedia cross-modal correlation discovery. In *SIGKDD*, 2004.

[16] J. Philbin and A. Zisserman. Object mining using a matching graph on very large image collections. In *ICVGIP*, 2008.

[17] A. Quattoni and A. Torralba. Recognizing indoor scenes. In *CVPR*, 2009.

[18] B. C. Russell, A. A. Efros, J. Sivic, W. T. Freeman, and A. Zisserman. Using multiple segmentations to discover objects and their extent in image collections. In *CVPR*, 2006.

[19] F. Schroff, A. Criminisi, and A. Zisserman. Harvesting image databases from the web. In *ICCV*, 2007.

[20] J. Shi and J. Malik. Normalized cuts and image segmentation. *IEEE Trans. on Pattern Analysis and Machine Intelligence*, 22(8):888–905, 2000.

[21] J. Sivic, B. C. Russell, A. A. Efros, A. Zisserman, and W. T. Freeman. Discovering objects and their location in images image features. In *ICCV*, 2005.

[22] A. Torralba, R. Fergus, and W. T. Freeman. 80 million tiny images: a large dataset for non-parametric object and scene recognition. *IEEE Trans. on Pattern Analysis and Machine Intelligence*, 30(11):1958–1970, 2008.

[23] U. von Luxburg. A tutorial on spectral clustering. *Statistics and Computing*, 17(4):395–416, 2007.

[24] J. Winn and N. Jojic. Locus: Learning object classes with unsupervised segmentation. In *ICCV*, 2005.

[25] D. Zhou, J. Weston, A. Gretton, O. Bousquet, and B. Schölkopf. Ranking on data manifolds. In *NIPS*, 2004.

